# Training Neural Networks with Deficient Data

**Volker Tresp**
Siemens AG
Central Research
81730 München
Germany
tresp@zfe.siemens.de

**Subutai Ahmad**
Interval Research Corporation
1801-C Page Mill Rd.
Palo Alto, CA 94304
ahmad@interval.com

**Ralph Neuneier**
Siemens AG
Central Research
81730 München
Germany
ralph@zfe.siemens.de

## Abstract

We analyze how data with uncertain or missing input features can be incorporated into the training of a neural network. The general solution requires a weighted integration over the unknown or uncertain input although computationally cheaper closed-form solutions can be found for certain Gaussian Basis Function (GBF) networks. We also discuss cases in which heuristical solutions such as substituting the mean of an unknown input can be harmful.

## 1 INTRODUCTION

The ability to learn from data with uncertain and missing information is a fundamental requirement for learning systems. In the "real world", features are missing due to unrecorded information or due to occlusion in vision, and measurements are affected by noise. In some cases the experimenter might want to assign varying degrees of reliability to the data.

In regression, uncertainty is typically attributed to the dependent variable which is assumed to be disturbed by additive noise. But there is no reason to assume that input features might not be uncertain as well or even missing competely.

In some cases, we can ignore the problem: instead of trying to model the relationship between the true input and the output we are satisfied with modeling the relationship between the uncertain input and the output. But there are at least two

reasons why we might want to explicitly deal with uncertain inputs. First, we might be interested in the underlying relationship between the true input and the output (e.g. the relationship has some physical meaning). Second, the problem might be non-stationary in the sense that for different samples different inputs are uncertain or missing or the levels of uncertainty vary. The naive strategy of training networks for all possible input combinations explodes in complexity and would require sufficient data for all relevant cases. It makes more sense to define one underlying true model and relate all data to this one model. Ahmad and Tresp (1993) have shown how to include uncertainty during recall under the assumption that the network approximates the "true" underlying function. In this paper, we first show how input uncertainty can be taken into account in the training of a feedforward neural network. Then we show that for networks of Gaussian basis functions it is possible to obtain closed-form solutions. We validate the solutions on two applications.

## 2  THE CONSEQUENCES OF INPUT UNCERTAINTY

Consider the task of predicting the dependent variable[1] $y \in \Re$ from the input vector $x \in \Re^M$ consisting of $M$ random variables. We assume that the input data $\{(x^k | k = 1, 2, ..., K\}$ are selected independently and that $P(x)$ is the joint probability distribution of $x$. Outputs $\{(y^k | k = 1, 2, ..., K\}$ are generated following the standard signal-plus-noise model

$$y^k = f(x^k) + \epsilon^k$$

where $\{\epsilon^k | k = 1, 2, ..., K\}$ denote zero-mean random variables with probability density $P_\epsilon(\epsilon)$. The best predictor (in the mean-squared sense) of $y$ given the input $x$ is the regressor defined by $E(y|x) = \int y \, P(y|x) \, dx = f(x)$, where $E$ denotes the expectation. Unbiased neural networks asymptotically ($K \to \infty$) converge to the regressor.

To account for uncertainty in the independent variable we assume that we do not have access to $x$ but can only obtain samples from another random vector $z \in \Re^M$ with

$$z^k = x^k + \delta^k$$

where $\{\delta^k | k = 1, 2, ..., K\}$ denote independent random vectors containing $M$ random variables with joint density $P_\delta(\delta)$.[2]

A neural network trained with data $\{(z^k, y^k) | k = 1, 2, ..., K\}$ approximates

$$E(y|z) = \frac{1}{P(z)} \int y \, P(y|x) \, P(z|x) \, P(x) \, dy dx = \frac{1}{P(z)} \int f(x) \, P_\delta(z - x) \, P(x) \, dx.$$

$$(1)$$

Thus, in general $E(y|z) \neq f(z)$ and we obtain a biased solution. Consider the case that the noise processes can be described by Gaussians $P_\epsilon(\epsilon) = G(\epsilon; 0, \sigma^y)$ and $P_\delta(\delta) = G(\delta; 0, \sigma)$ where, in our notation, $G(x; m, s)$ stands for

$$G(x; m, s) = \frac{1}{(2\pi)^{M/2} \prod_{j=1}^{M} s_j} \exp[-\frac{1}{2} \sum_{j=1}^{M} \frac{(x_j - m_j)^2}{s_j^2}]$$

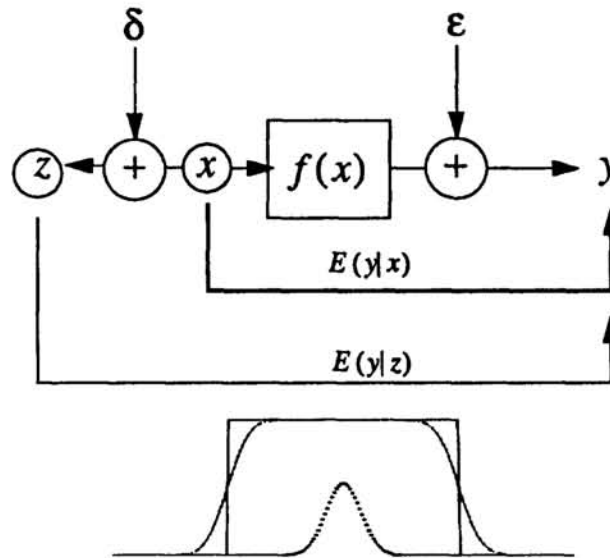

Figure 1: The top half of the figure shows the probabilistic model. In an example, the bottom half shows $E(y|x) = f(x)$ (continuous), the input noise distribution (dotted) and $E(y|z)$ (dashed).

where $m, s$ are vectors with the same dimensionality as $x$ (here $M$). Let us take a closer look at four special cases.

*Certain input.* If $\sigma = 0$ (no input noise), the integral collapses and $E(y|z) = f(z)$.

*Uncertain input.* If $P(x)$ varies much more slowly than $P(z|x)$, Equation 1 described the convolution of $f(x)$ with the noise process $P_\delta(z - x)$. Typical noise processes will therefore blur or smooth the original mapping (Figures 1). It is somewhat surprising that the error on the input results in a (linear) convolution integral. In some special cases we might be able to recover $f(x)$ from an network trained on deficient data by deconvolution, although one should use caution since deconvolution is very error sensitive.

*Unknown input.* If $\sigma_j \to \infty$ then the knowledge of $z_j$ does not give us any information about $x_j$ and we can consider the $j$th input to be unknown. Our formalism therefore includes the case of missing inputs as special case. Equation 1 becomes an integral over the unknown dimensions weighted by $P(x)$ (Figure 2).

*Linear approximation.* If the approximation

$$y^k = f(z^k - \delta^k) + \epsilon^k \approx f(z^k) - \sum_{j=1}^{M} \frac{\partial f}{\partial z_j}\Big|_{z^k} \delta_j^k + \epsilon^k \qquad (2)$$

is valid, the input noise can be transformed into output noise and $E(y|z) = f(z)$. This results can also be derived using Equation 1 if we consider that a convolution of a linear function with a symmetrical kernel does not change the function. This result tells us that if $f(x)$ is approximately linear over the range where $P_\delta(\delta)$ has significant

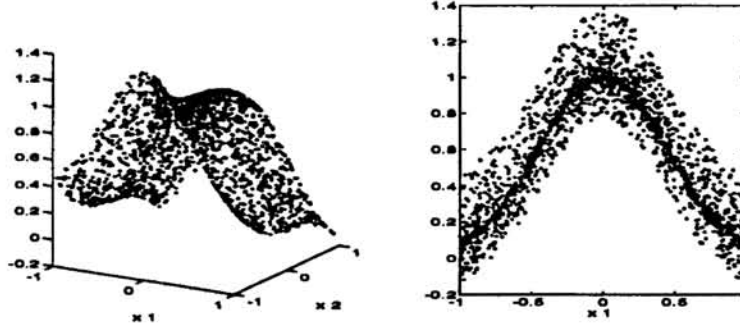

Figure 2: Left: samples $y^k = f(x_1^k, x_2^k)$ are shown (no output noise). Right: with one input missing, $P(y|x_1)$ appears noisy.

amplitude we can substitute the noisy input and the network will still approximate $f(x)$. Similarly, the mean $mean(x_j)$ of an unknown variable can be substituted for an unknown input, if $f(x)$ is linear and $x_j$ is independent of the remaining input variables. But in all those cases, one should be aware of the potentially large additional variance (Equation 2).

## 3   MAXIMUM LIKELIHOOD LEARNING

In this section, we demonstrate how deficient data can be incorporated into the training of feedforward networks. In a typical setting, we might have a number of complete data, a number of incomplete data and a number of data with uncertain features. Assuming independent samples and Gaussian noise, the log-likelihood $l$ for a neural network $NN_w$ with weight vector $w$ becomes

$$l = \sum_{k=1}^{K} \log P(z^k, y^k) = \sum_{k=1}^{K} \log \int G(y^k; NN_w(x), \sigma^y)\, G(z^k; x, \sigma^k)\, P(x)\, dx.$$

Note that now, the input noise variance is allowed to depend on the sample $k$. The gradient of the log-likelihood with respect to an arbitrary weight $w_i$ becomes[3]

$$\frac{\partial l}{\partial w_i} = \sum_{k=1}^{K} \frac{\partial \log P(z^k, y^k)}{\partial w_i} = \frac{1}{(\sigma^y)^2} \sum_{k=1}^{K} \frac{1}{P(z^k, y^k)} \times$$

$$\int (y^k - NN_w(x)) \frac{\partial NN_w(x)}{\partial w_i} G(y^k; NN_w(x), \sigma^y)\, G(z^k; x, \sigma^k)\, P(x)\, dx. \quad (3)$$

First, realize that for a certain sample $k$ ($\sigma^k \rightarrow 0$): $\partial \log P(z^k, y^k)/\partial w_i = (y^k - NN_w(z^k))/(\sigma^y)^2\, \partial NN_w(z^k)/\partial w_i$ which is the gradient used in normal back-propagation. For uncertain data, this gradient is replaced by an averaged gradient. The integral averages the gradient over possible true inputs $x$ weighted by the probability of $P(x|z^k, y^k) = P(z^k|x)\, P(y^k|x)\, P(x)/P(z^k, y^k)$. The term

$P(y^k|x) = G(y^k; NN_w(x), \sigma^y)$ is of special importance since it weights the gradient higher when the network prediction $NN_w(x)$ agrees with the target $y^k$. This term is also the main reason why heuristics such as substituting the mean value for a missing variable can be harmful: if, at the substituted input, the difference between network prediction and target is large, the error is also large and the data point contributes significantly to the gradient although it is very unlikely that the substitutes value was the true input.

In an implementation, the integral needs to be approximated by a finite sum (i. e. Monte-Carlo integration, finite-difference approximation etc.). In the experiment described in Figure 3, we had a 2-D input vector and the data set consisted of both complete data and data with one missing input. We used the following procedure

1. Train the network using the complete data. Estimate $(\sigma^y)^2$. We used $(\sigma^y)^2 \approx (E_C/(K-H))$, where $E_C$ is the training error after the network was trained with only the complete data, and $H$ is the number of hidden units in the network.

2. Estimate the input density $P(x)$ using Gaussian mixtures (see next section).

3. Include the incomplete training patterns in the training.

4. For every incomplete training pattern
   - Let $z_c^k$ be the certain input and let $z_u^k$ be the missing input, and $z^k = (z_c^k, z_u^k)$.
   - Approximate (assuming $-1/2 < x_j < 1/2$, the *hat* stands for estimate)

$$\frac{\partial \log P(z_c^k, y^k)}{\partial w_i} \approx \frac{1}{J} \frac{1}{(\sigma^y)^2} \frac{1}{\hat{P}(z_c^k, y^k)} \sum_{j=-J/2}^{J/2} ((y^k - NN_w(z_c^k, j/J)) \times$$

$$\frac{\partial NN_w(z_c^k, j/J)}{\partial w_i} G(y^k; NN_w(z_c^k, j/J), \sigma^y) \hat{P}(z_c^k, j/J)$$

where

$$\hat{P}(z_c^k, y^k) = \frac{1}{J} \sum_{j=-J/2}^{J/2} G(y^k; NN_w(z_c^k, j/J), \sigma^y) \hat{P}(z_c^k, j/J)$$

## 4   GAUSSIAN BASIS FUNCTIONS

The required integration in Equation 1 is computationally expensive and one would prefer closed form solutions. Closed form solutions can be found for networks which are based on Gaussian mixture densities.[4] Let's assume that the joint density is given by

$$P(x) = \sum_{i=1}^{N} G(x; c_i, s_i) P(\omega_i),$$

where $c_i$ is the location of the center of the $i$th Gaussian and and $s_{ij}$ corresponds to the width of the $i$th Gaussian in the $j$th dimension and $P(\omega_i)$ is the prior probability of $\omega_i$. Based on this model we can calculate the expected value of any unknown

[4]Gaussian mixture learning with missing inputs is also addressed by Ghahramani and Jordan (1993). See also their contribution in this volume.

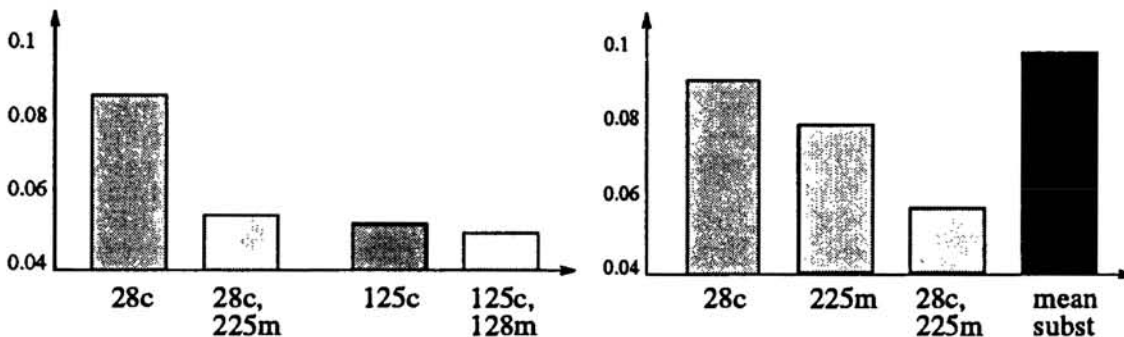

Figure 3: Regression. Left: We trained a *feedforward neural network* to predict the housing price from two inputs (average number of rooms, percent of lower status population (Tresp, Hollatz and Ahmad (1993)). The training data set contained varying numbers of complete data points (c) and data points with one input missing (m). For training, we used the method outlined in Section 3. The test set consisted of 253 complete data. The graph (vertical axis: generalization error) shows that by including the incomplete patterns in the training, the performance is significantly improved. Right: We approximated the joint density by a *mixture of Gaussians*. The incomplete patterns were included by using the procedure outlined in Section 4. The regression was calculated using Equation 4. As before, including the incomplete patterns in training improved the performance. Substituting the mean for the missing input (column on the right) on the other hand, resulted in worse performance than training of the network with only complete data.

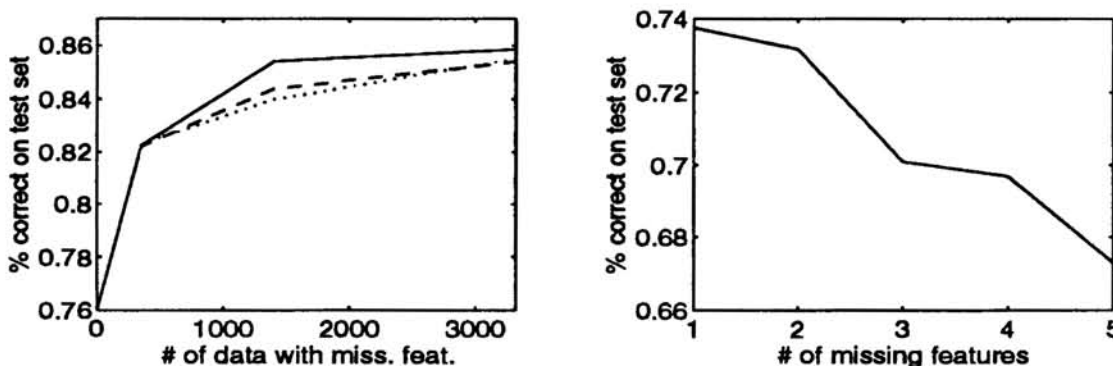

Figure 4: Left: Classification performance as a function of the number of missing features on the task of 3D hand gesture recognition using a *Gaussian mixtures classifier* (Equation 5). The network had 10 input units, 20 basis functions and 7 output units. The test set contained 3500 patterns. (For a complete description of the task see (Ahmad and Tresp, 1993).) Class-specific training with only 175 complete patterns is compared to the performance when the network is trained with an additional 350, 1400, and 3325 incomplete patterns. Either 1 input (continuous) or an equal number of 1-3 (dashed) or 1-5 (dotted) inputs where missing. The figure shows clearly that adding incomplete patterns to a data set consisting of only complete patterns improves performance. Right: the plot shows performance when the network is trained only with 175 incomplete patterns. The performance is relatively stable as the number of missing features increases.

variable $x^u$ from any set of known variables $x^n$ using (Tresp, Hollatz and Ahmad, 1993)

$$E(x^u|x^n) = \frac{\sum_{i=1}^{N} c_i^u G(x^n; c_i^n, s_i^n)\, P(\omega_i)}{\sum_{i=1}^{N} G(x^n; c_i^n, s_i^n)\, P(\omega_i)}. \tag{4}$$

Note, that the Gaussians are projected onto the known dimensions. The last equation describes the normalized basis function network introduced by Moody and Darken (1989).

Classifiers can be built by approximating the class-specific data distributions $P(x|class_i)$ by mixtures of Gaussians. Using Bayes formula, the posterior class probability then becomes

$$P(class_i|x) = \frac{P(class_i)P(x|class_i)}{\sum_j P(class_j)P(x|class_j)}. \tag{5}$$

We now assume that we do not have access to $x$ but to $z$ where, again, $P(z|x) = G(z; x, \sigma)$. The log-likelihood of the data now becomes

$$l = \sum_{k=1}^{K} \log \int \sum_{i=1}^{N} G(x; c_i, s_i)\, P(\omega_i)\, G(z^k; x, \sigma^k)\, dx = \sum_{k=1}^{K} \log \sum_{i=1}^{N} G(z^k; c_i, S_i^k)\, P(\omega_i)$$

where $(S_{ij}^k)^2 = s_{ij}^2 + (\sigma_j^k)^2$. We can use the EM approach (Dempster, Laird and Rubin, 1977) to obtain the following update equations. Let $c_{ij}, s_{ij}$ and $P(\omega_i)$ denote current parameter estimates and let $C_{ij}^k = (c_{ij}(\sigma_j^k)^2 + z_j s_{ij}^2)/(S_{ij}^k)^2$ and $D_{ij}^k = ((\sigma_j^k)^2 s_{ij}^2)/(S_{ij}^k)^2$. The new estimates (indicated by a *hat*) can be obtained using

$$\hat{P}(\omega_i|z^k) = \frac{G(z^k; c_i, S_i^k)\, P(\omega_i)}{\sum_{j=1}^{N} G(z^k; c_j, S_j^k)\, P(\omega_j)} \tag{6}$$

$$\hat{P}(\omega_i) = \frac{1}{K} \sum_{k=1}^{K} \hat{P}(\omega_i|z^k) \tag{7}$$

$$\hat{c}_{ij} = \frac{\sum_{k=1}^{K} C_{ij}^k\, \hat{P}(\omega_i|z^k)}{\sum_{k=1}^{K} \hat{P}(\omega_i|z^k)} \tag{8}$$

$$\hat{s}_{ij}^2 = \frac{\sum_{k=1}^{K} [D_{ij}^k + (C_{ij}^k - \hat{c}_{ij})^2]\, \hat{P}(\omega_i|z^k)}{\sum_{k=1}^{K} \hat{P}(\omega_i|z^k)}. \tag{9}$$

These equations can be solved by alternately using Equation 6 to estimate $\hat{P}(\omega_i|z^k)$ and Equations 7 to 9 to update the parameter estimates. If $\sigma^k = 0$ for all $k$ (only certain data) we obtain the well known EM equations for Gaussian mixtures (Duda and Hart (1973), page 200). Setting $\sigma_j^k = \infty$ represents the fact that the $j$th input is missing in the $k$th data point and $C_{ij}^k = c_{ij}$, $D_{ij}^k = s_{ij}^2$. Figure 3 and Figure 4 show experimental results for a regression and a classification problem.

## 5    EXTENSIONS AND CONCLUSIONS

We can only briefly address two more aspects. In Section 3 we only discussed regression. We can obtain similar results for classification problems if the cost-function is a log-likelihood function (e.g. the cross-entropy, the signal-plus-noise model is not appropriate). Also, so far we considered the true input to be unobserved *data*. Alternatively the true inputs can be considered unknown *parameters*. In this case, the goal is to substitute the maximum likely input for the unknown or noisy input. We obtain as log-likelihood function

$$l \propto \sum_{k=1}^{K} [-\frac{1}{2} \frac{(y^k - NN_w(x^k))^2}{(\sigma^y)^2} - \frac{1}{2} \sum_{j=1}^{M} \frac{(x_j^k - z_j^k)^2}{(\sigma_j^k)^2} + \log P(x^k)].$$

The learning procedure consists of finding optimal values for network weights $w$ and true inputs $x^k$.

## 6    CONCLUSIONS

Our paper has shown how deficient data can be included in network training. Equation 3 describes the solution for feedforward networks which includes a computationally expensive integral. Depending on the application, relatively cheap approximations might be feasible. Our paper hinted at possible pitfalls of simple heuristics. Particularly attractive are our results for Gaussian basis functions which allow closed-form solutions.

**References**

Ahmad, S. and Tresp, V. (1993). Some solutions to the missing feature problem in vision. In S. J. Hanson, J. D. Cowan and C. L. Giles, (Eds.), *Neural Information Processing Systems 5*. San Mateo, CA: Morgan Kaufmann.

Buntine, W. L. and Weigend, A. S. (1991). Bayesian Back-Propagation. *Complex systems*, Vol. 5, pp. 605-643.

Dempster, A. P., Laird, N. M. and Rubin, D. B. (1977). Maximum likelihood from incomplete data via the EM algorithm. *J. Royal Statistical Society Series B*, 39, pp. 1-38.

Duda, R. O. and Hart, P. E. (1973). *Pattern Classification and Scene Analysis*. John Wiley and Sons, New York.

Ghahramani, Z. and Jordan, M. I. (1993). *Function approximation via density estimation using an EM approach*. MIT Computational Cognitive Sciences, TR 9304.

Moody, J. E. and Darken, C. (1989). Fast learning in networks of locally-tuned processing units. *Neural Computation*, Vol. 1, pp. 281-294.

Tresp, V., Hollatz J. and Ahmad, S. (1993). Network structuring and training using rule-based knowledge. In S. J. Hanson, J. D. Cowan and C. L. Giles, (Eds.), *Neural Information Processing Systems 5*. San Mateo, CA: Morgan Kaufmann.

Tresp, V., Ahmad, S. and Neuneier, R. (1993). Uncertainty in the Inputs of Neural Networks. Presented at *Neural Networks for Computing 1993*.

## Footnotes

[1]Our notation does not distinguish between a random variable and its realization.

[2]At this point, we assume that $P_\delta$ is independent of $x$.

[3]This equation can also be obtained via the EM formalism. A similar equation was obtained by Buntine and Weigend (1991) for binary inputs.
